# Predicting Action Content On-Line and in Real Time before Action Onset — an Intracranial Human Study

**Uri Maoz**
California Institute of Technology
Pasadena, CA
urim@caltech.edu

**Shengxuan Ye**
California Institute of Technology
Pasadena, CA
sye@caltech.edu

**Ian Ross**
Huntington Hospital
Pasadena, CA
ianrossmd@aol.com

**Adam Mamelak**
Cedars-Sinai Medical Center
Los Angeles, CA
adam.mamelak@cshs.org

**Christof Koch**
California Institute of Technology
Pasadena, CA
Allen Institute for Brain Science
Seattle, WA
koch@klab.caltech.edu

## Abstract

The ability to predict action content from neural signals in real time before the action occurs has been long sought in the neuroscientific study of decision-making, agency and volition. On-line real-time (ORT) prediction is important for understanding the relation between neural correlates of decision-making and conscious, voluntary action as well as for brain-machine interfaces. Here, epilepsy patients, implanted with intracranial depth microelectrodes or subdural grid electrodes for clinical purposes, participated in a "matching-pennies" game against an opponent. In each trial, subjects were given a 5 s countdown, after which they had to raise their left or right hand immediately as the "go" signal appeared on a computer screen. They won a fixed amount of money if they raised a different hand than their opponent and lost that amount otherwise. The question we here studied was the extent to which neural precursors of the subjects' decisions can be detected in intracranial local field potentials (LFP) prior to the onset of the action.

We found that combined low-frequency (0.1–5 Hz) LFP signals from 10 electrodes were predictive of the intended left-/right-hand movements before the onset of the go signal. Our ORT system predicted which hand the patient would raise 0.5 s before the go signal with $68\pm3\%$ accuracy in two patients. Based on these results, we constructed an ORT system that tracked up to 30 electrodes simultaneously, and tested it on retrospective data from 7 patients. On average, we could predict the correct hand choice in 83% of the trials, which rose to 92% if we let the system drop 3/10 of the trials on which it was less confident. Our system demonstrates—for the first time—the feasibility of accurately predicting a binary action on single trials in real time for patients with intracranial recordings, well before the action occurs.

# 1  Introduction

The work of Benjamin Libet [1, 2] and others [3, 4] has challenged our intuitive notions of the relation between decision making and conscious voluntary action. Using electrocorticography (EEG), these experiments measured brain potentials from subjects that were instructed to flex their wrist at a time of their choice and note the position of a rotating dot on a clock when they felt the urge to move. The results suggested that a slow cortical wave measured over motor areas—termed "readiness potential" [5], and known to precede voluntary movement [6]—begins a few hundred milliseconds before the average reported time of the subjective 'urge' to move. This suggested that action onset and contents could be decoded from preparatory motor signals in the brain before the subject becomes aware of an intention to move and of the contents of the action. However, the readiness potential was computed by averaging over 40 or more trials aligned to movement onset after the fact. More recently, it was shown that action contents can be decoded using functional magnetic-resonance imaging (fMRI) several seconds before movement onset [7]. But, while done on a single-trial basis, decoding the neural signals took place off-line, after the experiment was concluded, as the sluggish nature of fMRI hemodynamic signals precluded real-time analysis. Moreover, the above studies focused on arbitrary and meaningless action—purposelessly raising the left or right hand—while we wanted to investigate prediction of reasoned action in more realistic, everyday situations with consequences for the subject.

Intracranial recordings are good candidates for single-trial, ORT analysis of action onset and contents [8, 9], because of the tight temporal pairing of LFP to the underlying neuronal signals. Moreover, such recordings are known to be cleaner and more robust, with signal-to-noise ratios up to 100 times larger than surface recordings like EEG [10, 11]. We therefore took advantage of a rare opportunity to work with epilepsy patients implanted with intracranial electrodes for clinical purposes. Our ORT system (Fig. 1) predicts, with far above chance accuracy, which one of two future actions is about to occur on this one trial and feeds the prediction back to the experimenter, all before the onset of the go signal that triggers the patient's movement (see Experimental Methods). We achieve relatively high prediction performance using only part of the data—learning from brain activity in past trials only (Fig. 2) to predict future ones (Fig. 3)—while still running the analysis quickly enough to act upon the prediction before the subject moved.

# 2  Experimental Methods

## 2.1  Subjects

Subjects in this experiment were 8 consenting intractable epilepsy patients that were implanted with intracranial electrodes as part of their presurgical clinical evaluation (ages 18–60, 3 males). They were inpatients in the neuro-telemetry ward at the Cedars Sinai Medical Center or the Huntington Memorial Hospital, and are designated with CS or HMH after their patient numbers, respectively. Six of them—P12CS, P15CS, P22CS and P29−31HMH were implanted with intracortical depth electrodes targeting their bilateral anterior-cingulate cortex, amygdala, hippocampus and orbitofrontal cortex. These electrodes had eight 40 μm microwires at their tips, 7 for recording and 1 serving as a local ground. Two patients, P15CS and P22CS, had additional microwires in the supplementary motor area. We utilized the LFP recorded from the microwires in this study. Two other patients, P16CS and P19CS, were implanted with an 8×8 subdural grid (64 electrodes) over parts of their temporal and prefrontal dorsolateral cortices. The data of one patient—P31HMH—was excluded because microwire signals were too noisy for meaningful analysis. The institutional review boards of Cedars Sinai Medical Center, the Huntington Memorial Hospital and the California Institute of Technology approved the experiments.

During the experiment, the subject sat in a hospital bed in a semi-inclined "lounge chair" position. The stimulus/analysis computer (bottom left of Fig. 4) displaying the game screen (bottom right inset of Fig. 4) was positioned to be easily viewable for the subject. When playing against the experimenter, the latter sat beside the bed. The response box was placed within easy reach of the subject (Fig. 4).

## 2.2  Experiment Design

As part of our focus on purposeful, reasoned action, we had the subjects play a matching-pennies game—a 2-choice version of "rock paper scissors"—either against the experimenter or against a computer. The subjects pressed down a button with their left hand and another with their right on a response box. Then, in each trial, there was a 5 s countdown followed by a go signal, after which they had to immediately lift one of their hands. It was agreed beforehand that the patient would win the trial if she lifted a different hand than her opponent, and lose if she raised the same hand as her opponent. Both players started off with a fixed amount of money, $5, and in each trial $0.10 was deducted from the loser and awarded to the winner. If a player lifted her hand before the go signal, did not lift her hand within 500 ms of the go signal, or lifted no hand or both hands at the go signal— an error trial—she lost $0.10 without her opponent gaining any money. The subjects were shown the countdown, the go signal, the overall score, and various instructions on a stimulus computer placed before them (Fig. 4). Each game consisted of 50 trials. If, at the end of the game, the subject had more money than her opponent, she received that money in cash from the experimenter.

Before the experimental session began, the experimenter explained the rules of the game to the subject, and she could practice playing the game until she was familiar with it. Consequently, patients usually made only few errors during the games ($<6\%$ of the trials). Following the tutorial, the subject played 1–3 games against the computer and then once against the experimenter, depending on their availability and clinical circumstances. The first 2 games of P12CS were removed because the subject tended to constantly raise the right hand regardless of winning or losing. Two patients, P15CS and P19CS, were tested in actual ORT conditions. In such sessions—3 games each—the subjects always played against the experimenter. These ORT games were different from the other games in two respects. First, a computer screen was placed behind the patient, in a location where she could not see it. Second, the experimenter was wearing earphones (Fig. 1,4). Half a second before go-signal onset, an arrow pointing towards the hand that the system predicted the experimenter had to raise to win the trial was displayed on that screen. Simultaneously, a monophonic tone was played in the experimenter's earphone ipsilateral to that hand. The experimenter then lifted that hand at the go signal (see Supplemental Movie).

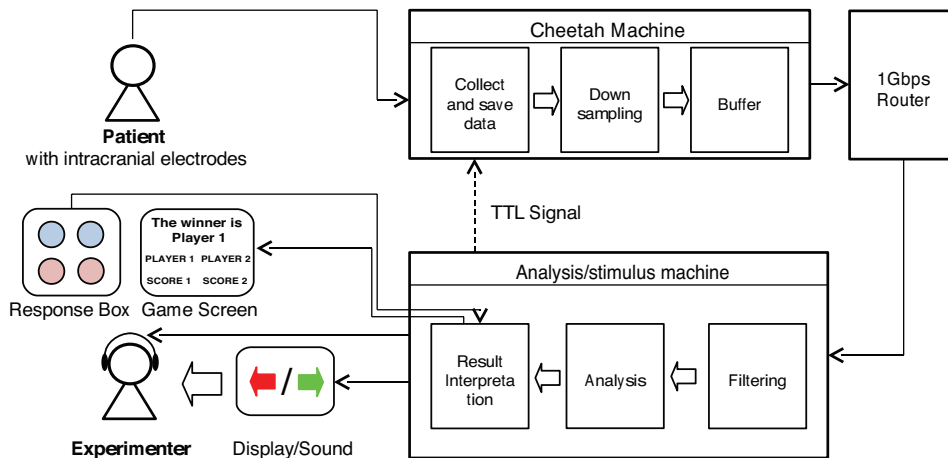

Figure 1: A schematic diagram of the on-line real-time (ORT) system. Neural signals flow from the patient through the Cheetah machine to the analysis/stimulus computer, which controls the input and output of the game and computes the prediction of the hand the patient would raise at the go signal. It displays it on a screen behind the patient and informs the experimenter which hand to raise by playing a tone in his ipsilateral ear using earphones.

# 3 The real-time system

## 3.1 Hardware and software overview

Neural data from the intracranial electrodes were transferred to a recording system (Neuralynx, Digital Lynx), where it was collected and saved to the local Cheetah machine, down sampled from 32 kHz to 2 kHz and buffered. The data were then transferred, through a dedicated 1 Gbps local-area network, to the analysis/stimulus machine. This computer first band-pass-filtered the data to the 0.1–5 Hz range (delta and lower theta bands) using a second-order zero-lag elliptic filter with an attenuation of 40 dB (cf. Figs. 2a and 2b). We found that this frequency range—generally comparable to that of the readiness potential—resulted in optimal prediction performance. It then ran the analysis algorithm (see below) on the filtered data. This computer also controlled the game screen, displaying the names of the players, their current scores and various instructions.

The analysis/stimulus computer further controlled the response box, which consisted of 4 LED-lit buttons. The buttons of the subject and her opponent flashed red or blue whenever she or her opponent won, respectively. Additionally, the analysis/stimulus computer sent a unique transistor-transistor logic (TTL) pulse whenever the game screen changed or a button was pressed on the response box, which synchronized the timing of these events with the LFP recordings. In real-time game sessions, the analysis/stimulus computer also displayed the appropriate arrow on the computer screen behind the subject and played the tone to the appropriate ear of the experimenter 0.5 s before go-signal onset (Figs. 1,4).

The analysis software was based on a machine-learning algorithm that trained on past-trials data to predict the current trial and is detailed below. The training phase included the first 70% of the trials, with the prediction carried out on the remaining 30% using the trained parameters, together with an online weighting system (see below). The system examined only neural activity, and had no access to the subject's left/right-choice history. After filtering all the training trials (Fig. 2b), the system found the mean and standard error over all leftward and rightward training trials, separately (Fig. 2c, left designated in red). It then found the electrodes and time windows where the left/right separation was high (Fig. 2d,e; see below), and trained the classifiers on these time windows (Fig. 2f–g). The best electrode/time-window/classifier (ETC) combinations were then used to predict the current trial in the prediction phase (Fig. 3). The number of ETCs that can be actively monitored is currently limited to 10 due to the computational power of the real-time system.

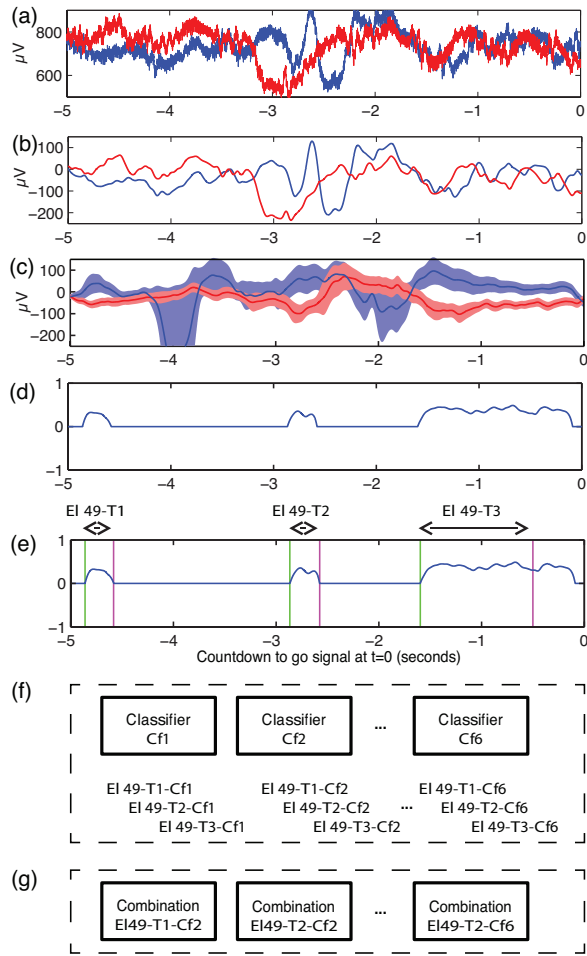

Figure 2: The ORT-system's training phase. Left (in red) and right (in blue) raw signals (a) are low-pass filtered (b). Mean±standard errors of signals preceeding left- and right-hand movments (c) are used to compute a left/right separability index (d), from which time windows with good separation are found (e). Seven classifiers are then applied to all the time windows (f) and the best electrode/time-window/classifier combinations are selected (g) and used in the prediction phase (Fig. 3).

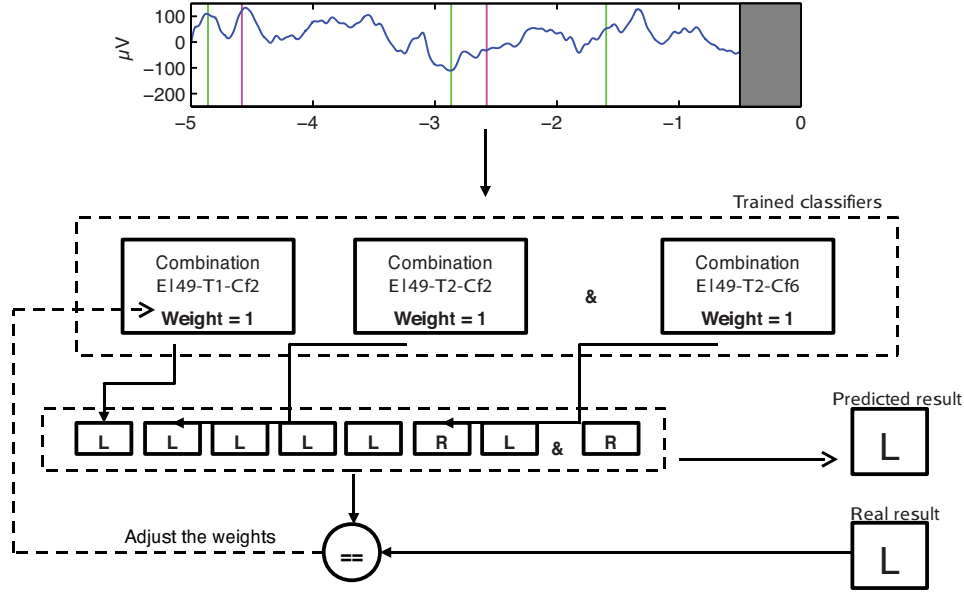

Figure 3: The ORT-system's prediction phase. A new signal—from 5 to 0.5 seconds before the go signal—is received in real time, and each electrode/time-window/classifier combination (ETC) classifies it as resulting in left- or right-hand movement. These predictions are then compared to the actual hand movement, with the weights associated with ETCs that correctly (incorrectly) predicted increasing (decreasing).

## 3.2 Computing optimal left/right-separating time windows

The algorithm focused on finding the time windows with the best left/right separation for the different recording electrodes over the training set (Fig. 2c–e). That is, we wanted to predict whether the signal $a_N(t)$ on trial $N$ will result in a leftward or rightward movement—i.e., whether the label of the $N^{\text{th}}$ trial will be Lt or Rt, respectively. For each electrode, we looked at the $N-1$ previous trials $a_1(t), a_2(t), \ldots, a_{N-1}(t)$, and their associated labels as $l_1, l_2, \ldots, l_{N-1}$. Now, let $L(t) = \{a_i(t) \,|\, l_i = \text{Lt}\}_{i=1}^{N-1}$ and $R(t) = \{a_i(t) \,|\, l_i = \text{Rt}\}_{i=1}^{N-1}$ be the set of previous leftward and rightward trials in the training set, respectively. Furthermore, let $L_m(t)$ ($R_m(t)$) and $L_s(t)$ ($R_s(t)$) be the mean and standard error of $L(t)$ ($R(t)$), respectively. We can now define the normalized relative left/right separation for each electrode at time $t$ (see Fig. 2d):

$$
\delta(t) = \begin{cases}
\dfrac{[L_m(t) - L_s(t)] - [R_m(t) + R_s(t)]}{L_m(t) - R_m(t)} & \text{if} \quad [L_m(t) - L_s(t)] - [R_m(t) + R_s(t)] > 0 \\[2ex]
-\dfrac{[R_m(t) - R_s(t)] - [L_m(t) + L_s(t)]}{R_m(t) - L_m(t)} & \text{if} \quad [R_m(t) - R_s(t)] - [L_m(t) + L_s(t)] > 0 \\[2ex]
0 & \text{otherwise}
\end{cases}
$$

Thus, $\delta(t) > 0$ ($\delta(t) < 0$) means that the leftward trials tend to be considerably higher (lower) than rightward trials for that electrode at time $t$, while $\delta(t) = 0$ suggests no left/right separation at time $t$. We define a consecutive time period of $|\delta(t)| > 0$ for $t <$ prediction time (the time before the go signal when we want the system to output a prediction; -0.5 s for the ORT trials) as a *time window* (Fig. 2e). After all time windows are found for all electrodes, time windows less than $M$ ms apart are combined into one. Then, for each time window from $t_1$ to $t_2$ we define $a = \int_{t_1}^{t_2} |\delta(t)| dt$. We then eliminate all time windows satisfying $a < A$. We found the values $M = 200$ ms and $A = 4,500\ \mu\text{V} \cdot \text{ms}$ to be optimal for real-time analysis. This resulted in 20–30 time windows over all 64 electrodes that we monitored.

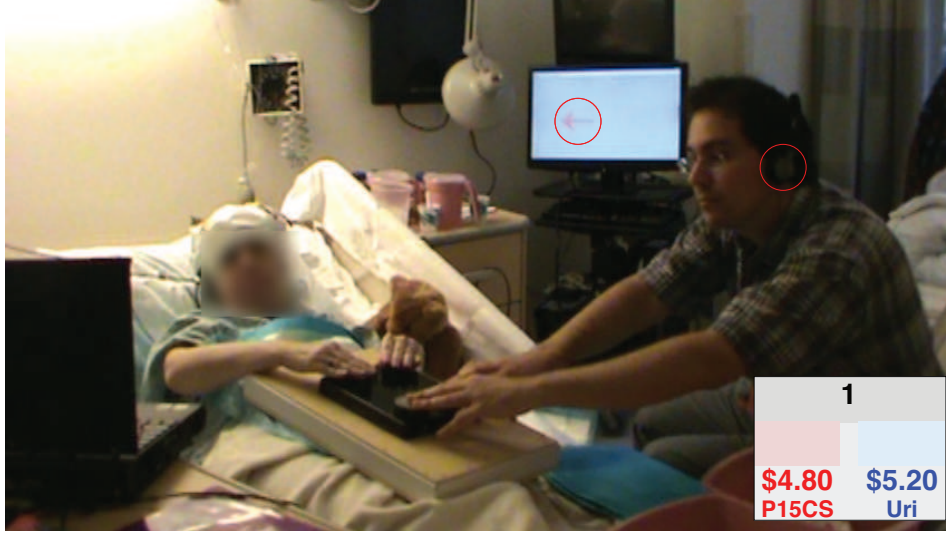

Figure 4: The experimental setup in the clinic. At 400 ms before the go signal, the patient and experimenter are watching the game screen (inset on bottom right) on the analysis/stimulus computer (bottom left) and still pressing down the buttons of the response box. The realtime system already computed a prediction, and thus displays an arrow on the screen behind the patient and plays a tone in the experimenter's ear ipsilateral to the hand it predicts he should raise to beat the patient (see Supplemental Movie).

### 3.3 Classifiers selection and ETC determination

We used ensemble learning with 7 types of relatively simple binary classifiers (due to real-time processing considerations) on every electrode's time windows (Fig. 2f). Classifiers **A** to **G** would classify $a_N(t)$ as *Lt* if:

**(A)** Defining $a_{N,M}$, $L_{m,M}$ and $R_{m,M}$ as $\sum a_N(t)$, $\sum L_m(t)$ and $\sum R_m(t)$ over time window $M$,

    (i) $\text{sign}(R_{m,M}) \neq \text{sign}(a_{N,M}) = \text{sign}(L_{m,M})$, or

    (ii) $\text{sign}(R_{m,M}) = \text{sign}(a_{N,M}) = \text{sign}(L_{m,M})$ and $|L_{m,M}| > |R_{m,M}|$, or

    (iii) $\text{sign}(R_m(t)) \neq \text{sign}(S_{N,M}) \neq \text{sign}(L_m(t))$ and $|L_{m,M}| < |R_{m,M}|$;

**(B)** $\left|\text{mean}(a_N(t)) - \text{mean}(L_m(t))\right| < \left|\text{mean}(a_N(t)) - \text{mean}(R_m(t))\right|$;

**(C)** $\left|\text{median}(a_N(t)) - \text{median}(L_m(t))\right| < \left|\text{median}(a_N(t)) - \text{median}(R_m(t))\right|$ over the time window;

**(D)** $\left|a_N(t) - L_m(t)\right|_{L2} < \left|a_N(t) - R_m(t)\right|_{L2}$ over the time window;

**(E)** $a_N(t)$ is convex/concave like $L_m(t)$ while $R_m(t)$ is concave/convex, respectively;

**(F)** Linear support-vector machine (SVM) designates it as so; and

**(G)** k-nearest neighbors (KNN) with Euclidean distance designates it as so.

Each classifier is optimized for certain types of features. To estimate how well its classification would generalize from the training to the test set, we trained and tested it using a 70/30 cross-validation procedure within the training set. We tested each classifier on every time window of every electrode, discarding those with accuracy $<0.68$, which left $12.0 \pm 1.6\%$ of the original $232 \pm 18$ ETCs, on average ($\pm$standard error). The training phase therefore ultimately output a set of $S$ binary ETC combinations (Fig. 2g) that were used in the prediction phase (Fig. 3).

### 3.4 The prediction-phase weighting system

In the prediction phase, each of the overall $S$ binary ETCs calculates a prediction, $c_i \in \{-1, 1\}$ (for right and left, respectively), independently at the desired prediction time. All classifiers are initially

given the same weight, $w_1 = w_2 = \cdots = w_S = 1$. We then calculate $\xi = \sum_{i=1}^{S} w_i \cdot c_i$ and predict left (right) if $\xi > d$ ($\xi < -d$), or declare it an undetermined trial if $-d < \xi < d$. Here $d$ is the drop-off threshold for the prediction. Thus the larger $d$ is, the more confident the system needs to be to make a prediction, and the larger the proportion of trials on which the system abstains—the drop-off rate. Weight $w_i$ associated with $\text{ETC}_i$ is increased (decreased) by 0.1 whenever $\text{ETC}_i$ predicts the hand movement correctly (incorrectly). A constantly erring ETC would therefore be associated with an increasingly small and then increasingly negative weight.

### 3.5 Implementation

The algorithm was implemented in MATLAB 2011a (MathWorks, Natick, MA) as well as in C++ on Visual Studio 2008 (Microsoft, Redmond, WA) for enhanced performance. The neural signals were collected by the Digital Lynx S system using Cheetah 5.4.0 (Neuralynx, Redmond, WA). The simulated-ORT system was also implemented in MATLAB 2011a. The simulated-ORT analyses carried out in this paper used real patient data saved on the Digital Lynx system.

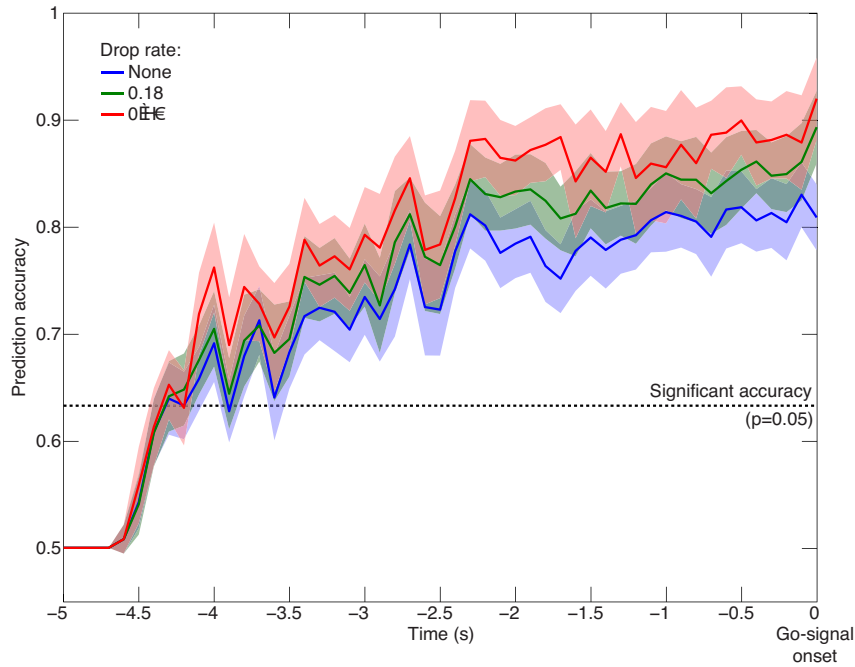

Figure 5: Across-subjects average of the prediction accuracy of simulated-ORT versus time before the go signal. The mean accuracies over time when the system predicts on every trial, is allowed to drop 19% or 30% of the trials, are depicted in blue, green and red, respectively ($\pm$standard error shaded). Values above the dashed horizontal line are significant at $p = 0.05$.

## 4 Results

We tested our prediction system in actual real time on 2 patients—P15CS and P19CS (a depth and grid patient, respectively), with a prediction time of 0.5 s before the go signal (see Supplementary Movie). Because of computational limitations, the ORT system could only track 10 electrodes with just 1 ETC per electrode in real time. For P15CS, we achieved an accuracy of $72\pm2\%$ ($\pm$standard error; accuracy = number of accurately predicted trials / [total number of trials - number of dropped trials]; $p = 10^{-8}$, binomial test) without modifying the weights online during the prediction (see Section 3.4). For P19CS we did not run patient-specific training of the ORT system, and used parameter values that were good on average over previous patients instead. The prediction accuracy was significantly above chance $63\pm2\%$ ($\pm$standard error; $p = 7 \cdot 10^{-4}$, binomial test). To understand how much we could improve our accuracy with optimized hardware/software, we ran the simulated-ORT at various prediction times along

the 5 s countdown leading to the go signal. We further tested 3 drop-off rates—0, 0.19 and 0.30 (Fig. 5; drop-off rate = number of dropped trials / total number of trials; these resulted from 3 drop-off thresholds—0, 0.1 and 0.2—respectively, see Section 3.4:). Running offline, we were able to track 20–30 ETCs, which resulted in considerably higher accuracies (Figs. 5,6).

Averaged over all subjects, the accuracy rose from about 65% more than 4 s before the go signal to 83–92% close to go-signal onset, depending on the allowed drop-off rate. In particular, we found that for a prediction time of 0.5 s before go-signal onset, we could achieve accuracies of 81±5% and 90±3% (±standard error) for P15CS and P19CS, respectively, with no drop off (Fig. 6). We also analyzed the weights that our weighting system assigned to the different ETCs. We found that the empirical distribution of weights to ETCs associated with classifiers A to G was, on average: 0.15, 0.12, 0.16, 0.22, 0.01, 0.26 and 0.07, respectively. This suggests that the linear SVM and L2-norm comparisons (of $a_N$ to $L_m$ and $R_m$) together make up nearly half of the overall weights attributed to the classifiers, while the current concave/convex measure is of little use as a classifier.

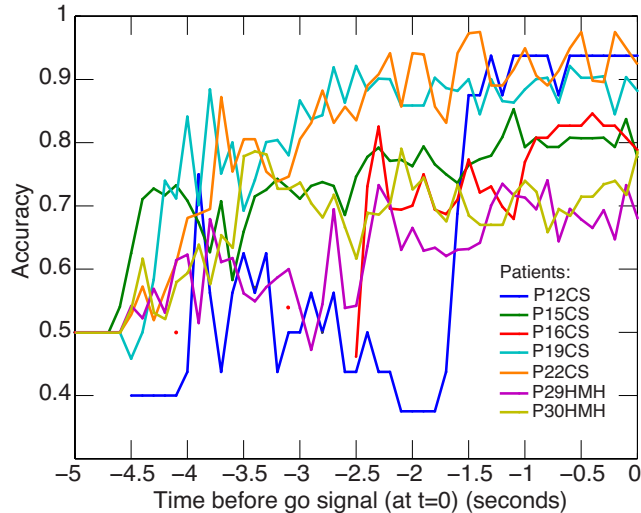

Figure 6: Simulated-ORT accuracy over time for individual patients with no drop off.

## 5  Discussion

We constructed an ORT system that, based on intracranial recordings, predicted which hand a person would raise well before movement onset at accuracies much greater than chance in a competitive environment. We further tested this system off-line, which suggested that with optimized hardware/software, such action contents would be predictable in real time at relatively high accuracies already several seconds before movement onset. Both our prediction accuracy and drop-off rates close to movement onset are superior to those achieved before movement onset with non-invasive methods like EEG and fMRI [7, 12–14]. Importantly, our subjects played a matching pennies game—a 2-choice version of rock-paper-scissors [15]—to keep their task realistic, with minor though real consequences, unlike the Libet-type paradigms whose outcome bears no consequences for the subjects. It was suggested that accurate online, real-time prediction before movement onset is key to investigating the relation between the neural correlates of decisions, their awareness, and voluntary action [16, 17]. Such prediction capabilities would facilitate many types of experiments that are currently infeasible. For example, it would make it possible to study decision reversals on a single-trial basis, or to test whether subjects can guess above chance which of their action contents are predictable from their current brain activity, potentially before having consciously made up their mind [16, 18]. Accurately decoding these preparatory motor signals may also result in earlier and improved classification for brain-computer interfaces [13, 19, 20]. The work we present here suggests that such ORT analysis might well be possible.

### Acknowledgements

We thank Ueli Rutishauser, Regan Blythe Towel, Liad Mudrik and Ralph Adolphs for meaningful discussions. This research was supported by the Ralph Schlaeger Charitable Foundation, Florida State University's "Big Questions in Free Will" initiative and the G. Harold & Leila Y. Mathers Charitable Foundation.

## References

[1] B. Libet, C. Gleason, E. Wright, and D. Pearl. Time of conscious intention to act in relation to onset of cerebral activity (readiness-potential): The unconscious initiation of a freely voluntary act. *Brain*, 106:623, 1983.

[2] B. Libet. Unconscious cerebral initiative and the role of conscious will in voluntary action. *Behavioral and brain sciences*, 8:529–539, 1985.

[3] P. Haggard and M. Eimer. On the relation between brain potentials and the awareness of voluntary movements. *Experimental Brain Research*, 126:128–133, 1999.

[4] A. Sirigu, E. Daprati, S. Ciancia, P. Giraux, N. Nighoghossian, A. Posada, and P. Haggard. Altered awareness of voluntary action after damage to the parietal cortex. *Nature Neuroscience*, 7:80–84, 2003.

[5] H. Kornhuber and L. Deecke. Hirnpotentiälanderungen bei Willkürbewegungen und passiven Bewegungen des Menschen: Bereitschaftspotential und reafferente Potentiale. *Pflügers Archiv European Journal of Physiology*, 284:1–17, 1965.

[6] H. Shibasaki and M. Hallett. What is the Bereitschaftspotential? *Clinical Neurophysiology*, 117:2341–2356, 2006.

[7] C. Soon, M. Brass, H. Heinze, and J. Haynes. Unconscious determinants of free decisions in the human brain. *Nature Neuroscience*, 11:543–545, 2008.

[8] I. Fried, R. Mukamel, and G. Kreiman. Internally generated preactivation of single neurons in human medial frontal cortex predicts volition. *Neuron*, 69:548–562, 2011.

[9] M. Cerf, N. Thiruvengadam, F. Mormann, A. Kraskov, R. Quian Quiorga, C. Koch, and I. Fried. On-line, voluntary control of human temporal lobe neurons. *Nature*, 467:1104–1108, 2010.

[10] T. Ball, M. Kern, I. Mutschler, A. Aertsen, and A. Schulze-Bonhage. Signal quality of simultaneously recorded invasive and non-invasive EEG. *Neuroimage*, 46:708–716, 2009.

[11] G. Schalk, J. Kubanek, K. Miller, N. Anderson, E. Leuthardt, J. Ojemann, D. Limbrick, D. Moran, L. Gerhardt, and J. Wolpaw. Decoding two-dimensional movement trajectories using electrocorticographic signals in humans. *Journal of Neural engineering*, 4:264, 2007.

[12] O. Bai, V. Rathi, P. Lin, D. Huang, H. Battapady, D. Y. Fei, L. Schneider, E. Houdayer, X. Chen, and M. Hallett. Prediction of human voluntary movement before it occurs. *Clinical Neurophysiology*, 122:364–372, 2011.

[13] O. Bai, P. Lin, S. Vorbach, J. Li, S. Furlani, and M. Hallett. Exploration of computational methods for classification of movement intention during human voluntary movement from single trial EEG. *Clinical Neurophysiology*, 118:2637–2655, 2007.

[14] U. Maoz, A. Arieli, S. Ullman, and C. Koch. Using single-trial EEG data to predict laterality of voluntary motor decisions. *Society for Neuroscience*, 38:289.6, 2008.

[15] C. Camerer. *Behavioral game theory: Experiments in strategic interaction.* Princeton University Press, 2003.

[16] J. D. Haynes. Decoding and predicting intentions. *Annals of the New York Academy of Sciences*, 1224:9–21, 2011.

[17] P. Haggard. Decision time for free will. *Neuron*, 69:404–406, 2011.

[18] J. D. Haynes. Beyond libet. In W. Sinnott-Armstrong and L. Nadel, editors, *Conscious will and responsibility*, pages 85–96. Oxford University Press, 2011.

[19] A. Muralidharan, J. Chae, and D. M. Taylor. Extracting attempted hand movements from EEGs in people with complete hand paralysis following stroke. *Frontiers in neuroscience*, 5, 2011.

[20] E. Lew, R. Chavarriaga, S. Silvoni, and J. R. Milln. Detection of self-paced reaching movement intention from EEG signals. *Frontiers in Neuroengineering*, 5:13, 2012.

